# SPEECH RECOGNITION: STATISTICAL AND NEURAL INFORMATION PROCESSING APPROACHES

John S. Bridle
Speech Research Unit and
National Electronics Research Initiative in Pattern Recognition
Royal Signals and Radar Establishment
Malvern UK

Automatic Speech Recognition (ASR) is an artificial perception problem: the input is raw, continuous patterns (no symbols!) and the desired output, which may be words, phonemes, meaning or text, is symbolic. The most successful approach to automatic speech recognition is based on *stochastic models*. A stochastic model is a theoretical system whose internal state and output undergo a series of transformations governed by probabilistic laws [1]. In the application to speech recognition the unknown patterns of sound are treated as if they were outputs of a stochastic system [18,2]. Information about the classes of patterns is encoded as the structure of these "laws" and the probabilities that govern their operation. The most popular type of SM for ASR is also known as a "hidden Markov model."

There are several reasons why the SM approach has been so successful for ASR. It can describe the shape of the spectrum, and has a principled way of describing temporal order, together with variability of both. It is compatible with the hierarchical nature of speech structure [20,18,4], there are powerful algorithms for decoding with respect to the model (recognition), and for adapting the model to fit significant amounts of example data (learning). Firm theoretical (mathematical) foundations enable extensions to be accommodated smoothly (e.g. [3]).

There are many deficiencies however. In a typical system the speech signal is first described as a sequence of acoustic vectors (spectrum cross sections or equivalent) at a rate of say 100 per second. The pattern is assumed to consist of a sequence of segments corresponding to discrete states of the model. In each segment the acoustic vectors are drawn from a distribution characteristic of the state, but otherwise independent of one another and of the states before and after. In some systems there is a controlled relationship between states and the *phonemes* or *phones* of speech science, but most of the properties and notions which speech scientists assume are important are ignored.

Most SM approaches are also deficient at a pattern-recognition theory level: The parameters of the models are usually adjusted (using the Baum-Welch re-estimation method [5,2]) so as to maximise the likelihood of the data given the model. This is the right thing to do if the form of the model is actually appropriate for the data, but if not the parameter-optimisation method needs to be concerned with

discrimination between classes (phonemes, words, meanings,...) [28,29,30].

A HMM recognition algorithm is designed to find the best explanation of the input in terms of the model. It tracks scores for all plausible current states of the generator and throws away explanations which lead to a current state for which there is a better explanation (Bellman's Dynamic Programming). It may also throw away explanations which lead to a current state much worse than the best current state (score pruning), producing a Beam Search method. (It is important to keep many hypotheses in hand, particularly when the current input is ambiguous.)

Connectionist (or "Neural Network") approaches start with a strong pre-conception of the types of process to be used. They can claim some legitimacy by reference to new (or renewed) theories of cognitive processing. The actual mechanisms used are usually simpler than those of the SM methods, but the mathematical theory (of what can be learnt or computed for instance) is more difficult, particularly for structures which have been proposed for dealing with temporal structure.

One of the dreams for connectionist approaches to speech is a network whose inputs accept the speech data as it arrives, it would have an internal state which contains all necessary information about the past input, and the output would be as accurate and early as it could be. The training of networks with their own dynamics is particularly difficult, especially when we are unable to specify what the internal state should be. Some are working on methods for training the fixed points of continuous-valued recurrent non-linear networks [15,16,27]. Prager [6] has attempted to train various types of network in a full state-feedback arrangement. Watrous [9] limits his recurrent connections to self-loops on hidden and output units, but even so the theory of such recursive non-linear filters is formidable.

At the other extreme are systems which treat a whole time-frequency-amplitude array (resulting from initial acoustic analysis) as the input to a network, and require a label as output. For example, the performance that Peeling et al. [7] report on multi-speaker small-vocabulary isolated word recognition tasks approach those of the best HMM techniques available on the same data. Invariance to temporal position was trained into the network by presenting the patterns at random positions in a fixed time-window. Waibel et al. [8] use a powerful compromise arrangement which can be thought of either as the replication of smaller networks across the time-window (a time-spread network [19]) or as a single small network with internal delay lines (a Time-Delay Neural Network [8]). There are no recurrent links except for trivial ones at the output, so training (using Backpropagation) is no great problem. We may think of this as a finite-impulse-response non-linear filter. Reported results on consonant discrimination are encouraging, and better than those of a HMM system on the same data. The system is insensitive to position by virtue of its construction.

Kohonen has constructed and demonstrated large vocabulary isolated word [12] and unrestricted vocabulary continuous speech transcription [13] systems which are inspired by neural network ideas, but implemented as algorithms more suitable for

current programmed digital signal processor and CPU chips. Kohonen's *phonotopic map* technique can be thought of as an unsupervised adaptive quantiser constrained to put its reference points in a non-linear low-dimensional sub-space. His *learning vector quantiser* technique used for initial labeling combines the advantages of the classic nearest-neighbor method and discriminant training.

Among other types of network which have been applied to speech we must mention an interesting class based not on correlations with weight vectors (dot-product) but on distances from reference points. *Radial Basis Function* theory [22] was developed for multi-dimensional interpolation, and was shown by Broomhead and Lowe [23] to be suitable for many of the jobs that feed-forward networks are used for. The advantage is that it is not difficult to find useful positions for the reference points which define the first, non-linear, transformation. If this is followed by a linear output transformation then the weights can be found by methods which are fast and straightforward. The reference points can be adapted using methods based on back-propagation. Related methods include potential functions [24], Kernel methods [25] and the modified Kanerva network [26].

There is much to be gained form a careful comparison of the theory of stochastic model and neural network approaches to speech recognition. If a NN is to perform speech decoding in a way anything like a SM algorithm it will have a state which is not just one of the states of the hypothetical generative model; the state must include information about the distribution of possible generator states given the pattern so far, and the state transition function must update this distribution depending on the current speech input. It is not clear whether such an internal representation and behavior can be 'learned' from scratch by an otherwise unstructured recurrent network.

Stochastic model based algorithms seem to have the edge at present for dealing with temporal sequences. Discrimination-based training inspired by NN techniques may make a significant difference in performance.

It would seem that the area where NNs have most to offer is in finding non-linear transformations of the data which take us to a space (perhaps related to formant or articulatory parameters) where comparisons are more relevant to phonetic decisions than purely auditory ones (e.g., [17,10,11]). The resulting transformation could also be viewed as a set of 'feature detectors'. Or perhaps the NN should deliver posterior probabilities of the states of a SM directly [14].

The art of applying a stochastic model or neural network approach is to choose a class of models or networks which is realistic enough to be likely to be able to capture the distinctions (between speech sounds or words for instance) and yet have a structure which makes it amenable to algorithms for building the detail of the models based on examples, and for interpreting particular unknown patterns. Future systems will need to exploit the regularities described by phonetics, to allow the construction of high-performance systems with large vocabularies, and their adaptation to the characteristics of each new user.

There is no doubt that the Stochastic model based methods work best at present, but current systems are generally far inferior to humans even in situations where the usefulness of higher-level processing in minimal. I predict that the next generation of ASR systems will be based on a combination of connectionist and SM theory and techniques, with mainstream speech knowledge used in a rather soft way to decide the structure. It should not be long before the distinction I have been making will disappear [29].

[1] D. R. Cox and H. D. Millar, "The Theory of Stochastic Processes", Methuen, 1965. pp. 721-741.

[2] S. E. Levinson, L. R. Rabiner and M. M. Sohndi, "An introduction to the application of the theory of probabilistic functions of a Markov process to automatic speech recognition", Bell Syst. Tech. J., vol. 62, no. 4, pp. 1035-1074, Apr. 1983.

[3] M. R. Russell and R. K. Moore, "Explicit modeling of state occupancy in hidden Markov models of automatic speech recognition". IEEE ICASSP-85.

[4] S. E. Levinson, "A unified theory of composite pattern analysis for automatic speech recognition', in F. Fallside and W. Woods (eds.), "Computer Speech Processing", Prentice-Hall, 1984.

[5] L. E. Baum, "An inequality and associated maximisation technique in statistical estimation of probabilistic functions of a Markov process", Inequalities, vol. 3, pp. 1-8, 1972.

[6] R. G. Prager et al., "Boltzmann machines for speech recognition", Computer Speech and Language, vol. 1., no. 1, 1986.

[7] S. M. Peeling, R. K. moore and M. J. Tomlinson, "The multi-layer perceptron as a tool for speech pattern processing research", Proc. Inst. Acoustics Conf. on Speech and Hearing, Windermere, November 1986.

[8] Waibel et al., ICASSP88, NIPS88 and ASSP forthcoming.

[9] R. L. Watrous, "Connectionist speech recognition using the Temporal Flow model", Proc. IEEE Workshop on Speech Recognition, Harriman NY, June 1988.

[10] I. S. Howard and M. A. Huckvale, "Acoustic-phonetic attribute determination using multi-layer perceptrons", IEEE Colloquium Digest 1988/11.

[11] M. A. Huckvale and I. S. Howard, "High performance phonetic feature analysis for automatic speech recognition", ICASSP89.

[12] T. Kohonen et al., "On-line recognition of spoken words from a large vocabulary", Information Sciences 33, 3-30 (1984).

[13] T. Kohonen, "The 'Neural' phonetic typewriter", IEEE Computer, March 1988.

[14] H. Bourlard and C. J. Wellekens, "Multilayer perceptrons and automatic speech recognition", IEEE First Intl. Conf. Neural Networks, San Diego, 1987.

[15] R. Rohwer and S. Renals, "Training recurrent networks", Proc. N'Euro-88, Paris, June 1988.

[16] L. Almeida, "A learning rule for asynchronous perceptrons with feedback in a combinatorial environment", Proc. IEEE Intl. Conf. Neural Networks, San Diego 1987.

[17] A. R. Webb and D. Lowe, "Adaptive feed-forward layered networks as pattern classifiers: a theorem illuminating their success in discriminant analysis", sub. to Neural Networks.

[18] J. K. Baker, "The Dragon system: an overview", IEEE Trans. ASSP-23, no. 1, pp. 24-29, Feb. 1975.

[19] J. S. Bridle and R. K. Moore, "Boltzmann machines for speech pattern processing", Proc. Inst. Acoust., November 1984, pp. 1-8.

[20] B. H. Repp, "On levels of description in speech research", J. Acoust. Soc. Amer. vol. 69 p. 1462-1464, 1981.

[21] R. A. Cole et al, "Performing fine phonetic distinctions: templates vs. features", in J. Perkell and D. H. Klatt (eds.), "Symposium on invariance and variability of speech processes", Hillsdale, NJ, Erlbaum 1984.

[22] M. J. D. Powell, "Radial basis functions for multi-variate interpolation: a review", IMA Conf. on algorithms for the approximation of functions and data, Shrivenham 1985.

[23] D. Broomhead and D. Lowe, "Multi-variable interpolation and adaptive networks", RSRE memo 4148, Royal Signals and Radar Est., 1988.

[24] M. A. Aizerman, E. M. Braverman and L. I. Rozonoer, "On the method of potential functions", Automatika i Telemekhanika, vol. 26 no. 11, pp. 2086-2088, 1964.

[25] Hand, "Kernel discriminant analysis", Research Studies Press, 1982.

[26] R. W. Prager and F. Fallside, "Modified Kanerva model for automatic speech recognition", submitted to Cmputer Speech and Language.

[27] F. J. Pineda, "Generalisation of backpropagation to recurrent neural networks", Physical Review Letters 1987.

[28] L. R. Bahl et al., Proc. ICASSP88, pp. 493-496.

[29] H. Bourlard and C. J. Wellekens, "Links between Markov models and multi-layer perceptrons", this volume.

[30] L. Niles, H. Silverman, G. Tajchman, M. Bush, "How limited training data can allow a neural network to out-perform an 'optimal' classifier", Proc. ICASSP89.